# Adaptive Caching by Refetching

**Robert B. Gramacy**[*]**, Manfred K. Warmuth**[*]**, Scott A. Brandt, Ismail Ari**[†]
Department of Computer Science, UCSC
Santa Cruz, CA 95064
{rbgramacy, manfred, scott, ari}@cs.ucsc.edu

## Abstract

We are constructing caching policies that have 13-20% lower miss rates than the best of twelve baseline policies over a large variety of request streams. This represents an improvement of 49–63% over Least Recently Used, the most commonly implemented policy. We achieve this not by designing a specific new policy but by using on-line Machine Learning algorithms to dynamically shift between the standard policies based on their observed miss rates. A thorough experimental evaluation of our techniques is given, as well as a discussion of what makes caching an interesting on-line learning problem.

## 1   Introduction

Caching is ubiquitous in operating systems. It is useful whenever we have a small, fast main memory and a larger, slower secondary memory. In file system caching, the secondary memory is a hard drive or a networked storage server while in web caching the secondary memory is the Internet. The goal of caching is to keep within the smaller memory data objects (files, web pages, etc.) from the larger memory which are likely to be accessed again in the near future. Since the future request stream is not generally known, heuristics, called *caching policies*, are used to decide which objects should be discarded as new objects are retained. More precisely, if a requested object already resides in the cache then we call it a *hit*, corresponding to a low-latency data access. Otherwise, we call it a *miss*, corresponding to a high-latency data access as the data must be fetched from the slower secondary memory into the faster cache memory. In the case of a miss, room must be made in the cache memory for the new object. To accomplish this a caching policy discards from the cache objects which it thinks will cause the fewest or least expensive future misses.

In this work we consider twelve baseline policies including seven common policies (RAND, FIFO, LIFO, LRU, MRU, LFU, and MFU), and five more recently developed and very successful policies (SIZE and GDS [CI97], GD* [JB00], GDSF and LFUDA [ACD+99]). These algorithms employ a variety of directly observable criteria including recency of access, frequency of access, size of the objects, cost of fetching the objects from secondary memory, and various combinations of these.

The primary difficulty in selecting the best policy lies in the fact that each of these policies may work well in different situations or at different times due to variations in workload,

[*]Partial support from NSF grant CCR 9821087
[†]Supported by Hewlett Packard Labs, Storage Technologies Department

system architecture, request size, type of processing, CPU speed, relative speeds of the different memories, load on the communication network, etc. Thus the difficult question is: In a given situation, which policy should govern the cache? For example, the request stream from disk accesses on a PC is quite different from the request stream produced by web-proxy accesses via a browser, or that of a file server on a local network. The relative performance of the twelve policies vary greatly depending on the application. Furthermore, the characteristics of a single request stream can vary temporally for a fixed application. For example, a file server can behave quite differently during the middle of the night while making tape archives in order to backup data, whereas during the day its purpose is to serve file requests to and from other machines and/or users. Because of their differing decision criteria, different policies perform better given different workload characteristics. The request streams become even more difficult to characterize when there is a hierarchy or a network of caches handling a variety of file-type requests. In these cases, choosing a fixed policy for each cache in advance is doomed to be sub-optimal.

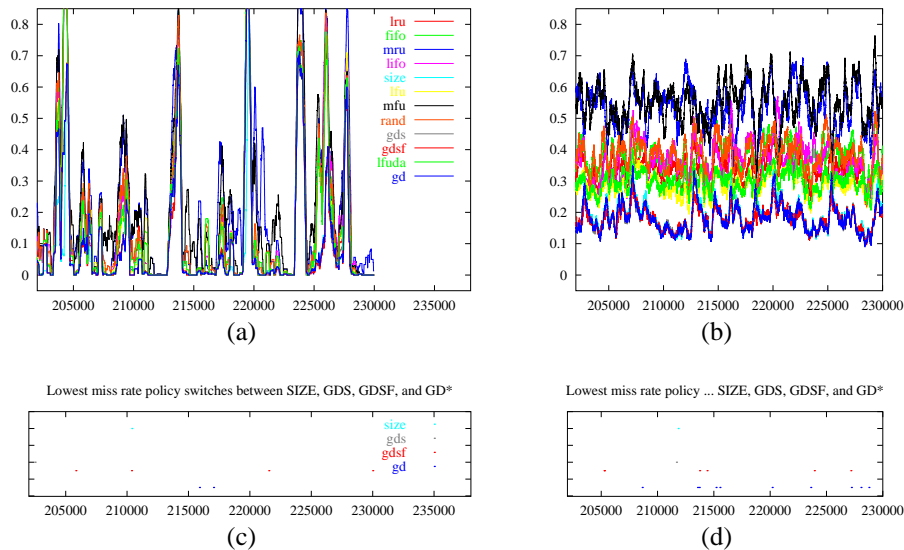

Figure 1: Miss rates ($y$ axis)of a) the twelve fixed policies (calculated w.r.t. a window of 300 requests) over 30,000 requests ($x$ axis), b) the same policies on a random permutation of the data set, c) and d) the policies with the lowest miss rates in the figures above.

The usual answer to the question of which policy to employ is either to select one that works well on average, or to select one that provides the best performance on some past workload that is believed to be representative. However, these strategies have two inherent costs. First, the selection (and perhaps tuning) of the single policy to be used in any given situation is done by hand and may be both difficult and error-prone, especially in complex system architectures with unknown and/or time-varying workloads. And second, the performance of the chosen policy with the best expected average case performance may in fact be worse than that achievable by another policy at any particular moment. Figure 1 (a) shows the hit rate of the twelve policies described above on a representative portion of one of our data sets (described below in Section 3) and Figure 1 (b) shows the hit rate of the same policies on a random permutation of the request stream. As can be clearly be seen, the miss rates on the permuted data set are quite different from those of the original data set, and it is this difference that our algorithms aim to exploit. Figures 1 (c) and (d) show which policy is best at each instant of time for the data segment and the permuted data segment. It is clear from these (representative) figures that the best policy changes over time.

To avoid the perils associated with trying to hand-pick a single policy, one would like to be able to automatically and dynamically select the best policy for any given situation. In other words, one wants a cache replacement policy which is "adaptive". In our Storage Systems Research Group, we have identified the need for such a solution in the context of complex network architectures and time-varying workloads and suggested a preliminary framework in which a solution could operate [AAG$^+$ar], but without giving specific algorithmic solutions to the adaptation problem. This paper presents specific algorithmic solutions that address the need identified in that work.

It is difficult to give a precise definition of "adaptive" when the data stream is continually changing. We use the term "adaptive" only informally and when we want to be precise we use off-line comparators to judge the performance of our on-line algorithms, as is commonly done in on-line learning [LW94, CBFH$^+$97, KW97]. An on-line algorithm is called *adaptive* if it performs well when measured up against off-line comparators. In this paper we use two off-line comparators: *BestFixed* and *BestShifting(K)*. Best-

Fixed is the *a posteriori* selected policy with the lowest miss rate on the entire request stream for our twelve policies. BestShifting($K$) considers all possible partitions of the request stream into at most $K$ segments along with the best policy for each segment. BestShifting($K$) chooses the partition with the lowest total miss rate over the entire dataset and can be computed in time $O(TKN)$ using dynamic programming. Here $T$ is the total number of requests, $K$ a bound on the number of segments, and $N$ the number of base-line policies. Figure 2

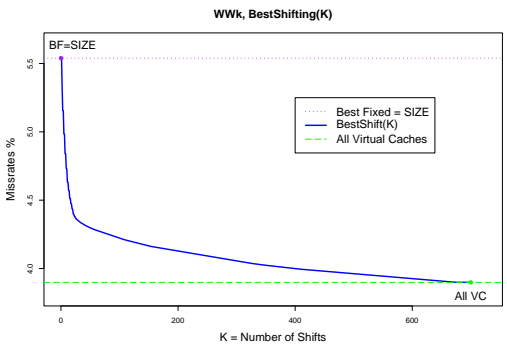

**Figure 2:** Optimal offline comparators. AllVC = $\lim_{K \to \infty}$ BestShifting($K$).

shows graphically each of the comparators mentioned above. Notice that BestFixed $\equiv$ BestShifting(1), and that most of the advantage of shifting policies occurs with relatively few shifts ($K \approx 50$ shifts in roughly 300,000 requests).

Rather than developing a new caching policy (well-plowed ground, to say the least), this paper uses a *master policy* to dynamically determine the success rate of all the other policies and switch among them based on their relative performance on the current request stream. We show that with no additional fetches, the master policy works about as well as BestFixed. We define a *refetch* as a fetch of a previously seen object that was favored by the current policy but discarded from the real cache by a previously active policy. With refetching, it can outperform BestFixed. In particular, when all required objects are refetched instantly, this policy has a 13-20% lower miss rate than BestFixed, and almost the same performance as BestShifting($K$) for modest $K$. For reference, when compared with LRU, this policy has a 49-63% lower miss rate. Disregarding misses on objects never seen before (*compulsory* misses), the performance improvements are even greater.

Because refetches themselves potentially costly, it is important to note that they can be done in the background. Our preliminary experiments show this to be both feasible and effective, capturing most of the advantage of instant refetching. A more detailed discussion of our results is given in Section 3

## 2   The Master Policy

We seek to develop an on-line master policy that determines which of a set of baseline policies should govern the real cache at any time. Appropriate switch points need to be found and switches must be facilitated. Our key idea is "virtual caches". A *virtual cache* simulates the operation of a single baseline policy. Each virtual cache records a few bytes of metadata about each object in its cache: ID, size, and calculated priority. Object data is only kept in the real cache, making the cost of maintaining the virtual caches negligible[1]. Via the virtual caches, the master policy can observe the miss rates of each policy on the actual request stream in order to determine their performance on the current workload.

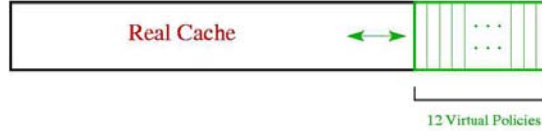

**Figure 3:**   Virtual caches embedded in the cache memory.

To be fair, virtual caches reside in the memory space which could have been used to cache real objects, as is illustrated in Figure 3. Thus, the space used by the real cache is reduced by the space occupied by the virtual caches. We set the virtual size of each virtual cache equal to the size of the full cache. The caches used for computing the comparators BestFixed and BestShifting($K$) are based on caches of the full size.

A simple heuristic the master policy can use to choose which caching policy should control at any given time is to continuously monitor the number of misses incurred by each policy in a past window of, for example, 300 requests (depicted in Figure 1 (a)). The master policy then gives control of the real cache to the policy with the least misses in this window (shown in Figure 1 (c)). While this works well in practice, maintaining such a window for many fixed policies is expensive, further reducing the space for the real cache. It is also hard to tune the window size. A better master policy keeps just one weight $w_i$ for each policy (non-negative and summing to one) which represents an estimate of its current relative performance. The master policy is always governed by the policy with the maximum weight[2].

Weights are updated by using the combined *loss* and *share* updates of Herbster and Warmuth [HW98] and Bousquet and Warmuth [BW02] from the expert framework [CBFH+97] for on-line learning. Here the experts are the caching policies. This technique is preferred to the window-based master policy because it uses much less memory, and because the parameters of the weight updates are easier to tune than the window size. This also makes the resulting master policy more robust (not shown).

### 2.1   The Weight Updates

Updating the weight vector $(w_1, \ldots, w_{12})$ after each trial is a two-part process. First, the weights of all policies that missed the new request are *multiplied* by a factor $\beta \in (0, 1)$ and then renormalized. We call this the *loss update*. Since the weights are renormalized, they remain unchanged if all policies miss the new request. As noticed by Herbster and Warmuth [HW98], multiplicative updates drive the weights of poor experts to zero so quickly that it becomes difficult for them to *recover* if their experts subsequently start doing well.

Therefore, the second *share update* prevents the weights of experts that did well in the past from becoming too small, allowing them to recover quickly, as shown in Figure 4. Figure 1(a) shows the current absolute performance of the policies in a rolling window ($W = 300$), whereas Figure 4 depicts relative performance and shows how the policies compete over time. (Recall that the policy with the highest weight always controls the real cache).

There are a number of share updates [HW98, BW02] with various recovery properties. We chose the FIXED SHARE TO UNIFORM PAST (FSUP) update because of its simplicity and efficiency. Note that the loss bounds proven in the expert framework for the combined loss and share update do not apply in this context. This is because we use the mixture weights only to select the best policy. However, our experimental results suggest that we are exploiting the

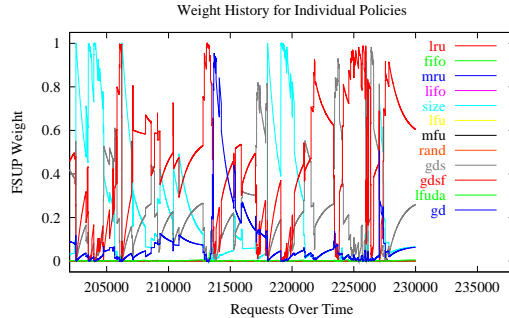

**Figure 4:** Weights of baseline policies.

recovery properties of the combined update that are discussed extensively by Bousquet and Warmuth [BW02].

Formally, for each trial $t$, the loss update is

$$w_{t,i}^m = \frac{w_{t,i}\beta^{\text{miss}_{t,i}}}{Z_{t+1}}, \quad Z_{t+1} = \sum_{i=1}^{N} w_{t,i}\beta^{\text{miss}_{t,i}}, \quad \text{for } i = 1, \ldots, N,$$

where $\beta$ is a parameter in $(0,1)$ and $\text{miss}_{t,i}$ is 1 if the $t$-th object is missed by policy $i$ and 0 otherwise. The initial distribution is uniform, i.e. $w_{1,i} = 1/N$. The Fixed-Share to Uniform Past update mixes the current weight vector with the past average weight vector $\mathbf{r}_t = \sum_{q=1}^{t} \mathbf{w}_q^m / t$, which is easy to maintain:

$$\mathbf{w}_{t+1} = (1 - \alpha)\, \mathbf{w}_t^m + \alpha\, \mathbf{r}_{t-1},$$

where $\alpha$ is a parameter in $(0,1)$. A small $\beta$ parameter causes high weight to decay quickly if its corresponding policy starts incurring more misses than other policies with high weights. The higher the $\alpha$ the more quickly past good policies will recover. In our experiments we used $\beta = .37$ and $\alpha = .005$.

## 2.2 Demand vs. Instantaneous Rollover

When space is needed to cache a new request, the master policy discards objects not present in the governing policy's virtual cache[3]. This causes the content of the real cache to "roll over" to the content of the current governing virtual cache. We call this *demand rollover* because objects in the governing virtual cache are refetched into the real cache on demand. While this master policy works almost as well as BestFixed, we were not satisfied and wanted to do as well as BestShifting($K$) (for a reasonably large bound $K$ on the number of segments). We noticed that the content of the real cache *lagged behind* the content of the governing virtual cache and had more misses, and conjectured that "quicker" rollover strategies would improve overall performance.

Our search for a better master policy began by considering an extreme and unrealistic rollover strategy that assures no lag time: After each switch *instantaneously* refetch all

the objects in the new governing virtual cache that were not retained in the real cache. We call this refetching policy *instantaneous rollover*. By appropriate tuning of the update parameters $\beta$ and $\alpha$ the number of instantaneous rollovers can be kept reasonably small and the miss rates of our master policy are almost as good as BestShifting($K$) for $K$ much larger than the actual number of shifts used on-line. Note that the comparator BestShifting($K$) is also not penalized for its instantaneous rollovers. While this makes sense for defining a comparator, we now give more realistic rollover strategies that reduce the lag time.

### 2.3 Background Rollover

Because instantaneous rollover immediately refetches everything in the governing virtual cache that is not already in the real cache, it may cause a large number of refetches even when the number of policy switches is kept small. If all refetches are counted as misses, then the miss rate of such a master policy is comparable to that of BestFixed. The same holds for BestShifting. However, from a user perspective, refetching is advantageous because of the latency advantage gained by having required objects in memory before they are needed. And from a system perspective, refetches can be "free" if they are done when the system is idle. To take advantage of these "free" refetches, we introduce the concept of *background rollover*. The exact criteria for when to refetch each missing object will depend heavily on the system, workload, and expected cost and benefit of each object. To characterize the performance of background rollover without addressing these architectural details, the following background refetching strategies were examined: 1 refetch for every cache miss; 1 for every hit; 1 for every request; 2 for every request; 1 for every hit and 5 for every miss, etc. Each background technique gave fewer misses than BestFixed, approaching and nearly matching the performance obtained by the master policy using instantaneous rollover. Of course, techniques which reduce the number of policy switches (by tuning $\beta$ and $\alpha$) also reduce the number of refetches. Figure 5 compares the performance of each master policy with that of BestFixed and shows that the three master policies almost always outperform BestFixed.

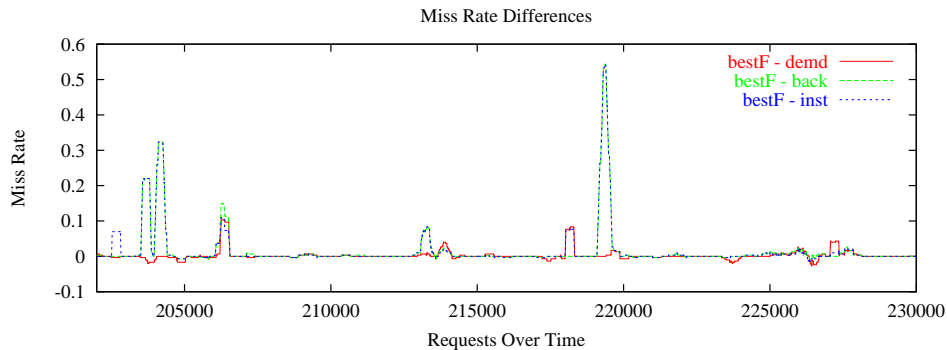

Figure 5: BestFixed - P, where P ∈ {Instantaneous, Demand, and Background Rollover 2}. The baseline ($x = 0$) is BestFixed. Deviations from the baseline $x = 0$ show how the performance of our on-line shifting policies differ in miss rate. Above (Below) $x = 0$ corresponds to fewer (more) misses than BestFixed.

## 3  Data and Results

Figure 6 shows how the master policy with instantaneous rollover (labeled 'roll') "tracks" the baseline policy with the lowest miss rate over the representative data segment used in previous figures. Figure 7 shows the performance of our master policies with respect to BestFixed, BestShifting($K$), and LRU. It shows that demand rollover does slightly worse than BestFixed, while background 1 (1 refetch every request) and background 2 (1 refetch

every hit and 5 every miss) do better than BestFixed and almost as well as instantaneous, which itself does almost as well as BestShifting. All of the policies do significantly better than LRU. Discounting the compulsory misses, our best policies have ∼1/3 fewer "real" misses than BestFixed and ∼1/2 the "real" misses of LRU.

Figure 8 summarizes the performance of our algorithms over three large datasets. These were gathered using Carnegie Mellon University's DFSTrace system [MS96] and had durations ranging from a single day to over a year. The traces we used represent a variety of workloads including a personal workstation (Work-Week), a single user (User-Month), and a remote storage system with a large number of clients, filtered by LRU on the clients' local caches (Server-Month-LRU). For each data set, the table shows the number of requests, % of requests skipped (size > cache size), number of compulsory misses of objects not previously seen, and the number of rollovers. For each policy (including BestShifting($K$)), the table shows miss rate, and % improvement over BestFixed (labeled '% <BF') and LRU. In each case all 12 virtual caches consumed on average less than 2% of the real cache space. We fixed $\beta = .37$, $\alpha = .005$ for all experiments. As already mentioned, BestShifting($K$) is never penalized for rollovers.

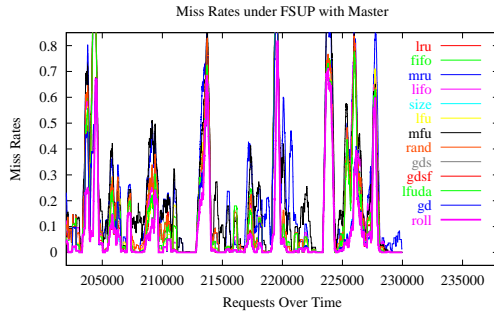

Figure 6: "Tracking" the best policy.

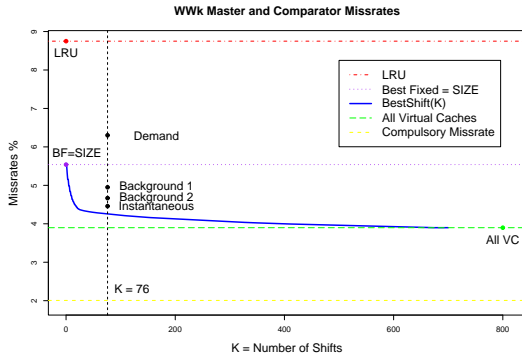

Figure 7: Online shifting policies against offline comparators and LRU for Work-Week dataset.

|  | Dataset | | |
|---|---|---|---|
|  | Works Week | User Month | Server Month LRU |
| #Requests | 138k | 382k | 48k |
| Cache size | 900KB | 2MB | 4MB |
| %Skipped | 6.5% | 12.8% | 15.7% |
| # Compuls | 0.020 | 0.015 | 0.152 |
| # Shifts | 88 | 485 | 93 |
| **LRU** | | | |
| Miss Rate | 0.088 | 0.076 | 0.450 |
| **BestFixed** | | | |
| Policy | SIZE | GDS | GDSF |
| Miss Rate | 0.055 | 0.075 | 0.399 |
| % <LRU | 36.8% | 54.7% | 54.2% |
| **Demand** | | | |
| Miss Rate | 0.061 | 0.076 | 0.450 |
| **% <BestF** | **-9.6%** | **-0.5%** | **-12.8%** |
| % <LRU | 30.9% | 54.4% | 48.5% |
| **Backgrnd 1** | | | |
| Miss Rate | 0.053 | 0.068 | 0.401 |
| **% <BestF** | **5.1%** | **9.8%** | **-0.7%** |
| % <LRU | 40.1% | 59.4% | 55.5% |
| **Backgrnd 2** | | | |
| Miss Rate | 0.047 | 0.067 | 0.349 |
| **% <BestF** | **15.4%** | **11.9%** | **12.4%** |
| % <LRU | 46.6% | 60.1% | 60.3% |
| **Instant** | | | |
| Miss Rate | 0.044 | 0.065 | 0.322 |
| **% <BestF** | **19.7%** | **13.4%** | **19.3%** |
| % <LRU | 49.2% | 60.8% | 63% |
| **BestShifting** | | | |
| Miss Rate | 0.042 | 0.039 | 0.312 |
| **% <BestF** | **23.6%** | **48.0%** | **21.8%** |
| % <LRU | 52.2% | 48.7% | 30.1% |

Figure 8: Performance Summary.

# 4 Conclusion

Operating systems have many hidden parameter tweaking problems which are ideal applications for on-line Machine Learning algorithms. These parameters are often set to values

which provide good average case performance on a test workload. For example, we have identified candidate parameters in device management, file systems, and network protocols. Previously the on-line algorithms for predicting as well as the best shifting expert were used to tune the time-out for spinning down the disk of a PC [HLSS00]. In this paper we use the weight updates of these algorithms for dynamically determining the best caching policy. This application is more elaborate because we needed to actively gather performance information about the caching policies via virtual caches. In future work we plan to do a more thorough study of feasibility of background rollover by building actual systems.

**Acknowledgements:** Thanks to David P. Helmbold for an efficient dynamic programming approach to BestShifting($K$), Ahmed Amer for data, and Ethan Miller many helpful insights.

## Footnotes

[1]As an additional optimization, we record the id and size of each object only once, regardless of the number of virtual caches it appears in.

[2]This can be sub-optimal in the worst case since it is always possible to construct a data stream where two policies switch back and forth after each request. However, real request streams appear to be divided into segments that favor one of the twelve policies for a substantial number of requests (see Figure 1).

[3]We update the virtual caches before the real cache, so there are always objects in the real cache that are not in the governing virtual cache when the master policy goes to find space for a new request.

# References

[AAG$^+$ar] Ismail Ari, Ahmed Amer, Robert Gramacy, Ethan Miller, Scott Brandt, and Darrell D. E. Long. ACME: Adaptive caching using multiple experts. In *Proceedings of the 2002 Workshop on Distributed Data and Structures (WDAS 2002)*. Carleton Scientific, (to appear).

[ACD$^+$99] Martin Arlitt, Ludmilla Cherkasova, John Dilley, Rich Friedrich, and Tai Jin. Evaluating content management techniques for Web proxy caches. In *Proceedings of the Workshop on Internet Server Performance (WISP99)*, May 1999.

[BW02] O. Bousquet and M. K. Warmuth. Tracking a small set of experts by mixing past posteriors. *J. of Machine Learning Research*, 3(Nov):363–396, 2002. Special issue for COLT01.

[CBFH$^+$97] N. Cesa-Bianchi, Y. Freund, D. Haussler, D. P. Helmbold, R. E. Schapire, and M. K. Warmuth. How to use expert advice. *Journal of the ACM*, 44(3):427–485, 1997.

[CI97] Pei Cao and Sandy Irani. Cost-aware WWW proxy caching algorithms. In *Proceedings of the 1997 Usenix Symposium on Internet Technologies and Systems (USITS-97)*, 1997.

[HLSS00] David P. Helmbold, Darrell D. E. Long, Tracey L. Sconyers, and Bruce Sherrod. Adaptive disk spin-down for mobile computers. *ACM/Baltzer Mobile Networks and Applications (MONET)*, pages 285–297, 2000.

[HW98] M. Herbster and M. K. Warmuth. Tracking the best expert. *Journal of Machine Learning*, 32(2):151–178, August 1998. Special issue on concept drift.

[JB00] Shudong Jin and Azer Bestavros. Greedydual* web caching algorithm: Exploiting the two sources of temporal locality in web request streams. Technical Report 2000-011, 4, 2000.

[KW97] J. Kivinen and M. K. Warmuth. Additive versus exponentiated gradient updates for linear prediction. *Information and Computation*, 132(1):1–64, January 1997.

[LW94] N. Littlestone and M. K. Warmuth. The weighted majority algorithm. *Information and Computation*, 108(2):212–261, 1994.

[MS96] Lily Mummert and Mahadev Satyanarayanan. Long term distributed file reference tracing: Implementation and experience. *Software - Practice and Experience (SPE)*, 26(6):705–736, June 1996.
